# Bayesian Sets

**Zoubin Ghahramani**[*] and   **Katherine A. Heller**
Gatsby Computational Neuroscience Unit
University College London
London WC1N 3AR, U.K.
{zoubin,heller}@gatsby.ucl.ac.uk

## Abstract

Inspired by "Google™ Sets", we consider the problem of retrieving items from a concept or cluster, given a query consisting of a few items from that cluster. We formulate this as a Bayesian inference problem and describe a very simple algorithm for solving it. Our algorithm uses a model-based concept of a cluster and ranks items using a score which evaluates the marginal probability that each item belongs to a cluster containing the query items. For exponential family models with conjugate priors this marginal probability is a simple function of sufficient statistics. We focus on sparse binary data and show that our score can be evaluated exactly using a single sparse matrix multiplication, making it possible to apply our algorithm to very large datasets. We evaluate our algorithm on three datasets: retrieving movies from EachMovie, finding completions of author sets from the NIPS dataset, and finding completions of sets of words appearing in the Grolier encyclopedia. We compare to Google™ Sets and show that Bayesian Sets gives very reasonable set completions.

## 1   Introduction

What do Jesus and Darwin have in common? Other than being associated with two different views on the origin of man, they also have colleges at Cambridge University named after them. If these two names are entered as a query into Google™ Sets (`http://labs.google.com/sets`) it returns a list of other colleges at Cambridge.

Google™ Sets is a remarkably useful tool which encapsulates a very practical and interesting problem in machine learning and information retrieval.[1] Consider a universe of items $\mathcal{D}$. Depending on the application, the set $\mathcal{D}$ may consist of web pages, movies, people, words, proteins, images, or any other object we may wish to form queries on. The user provides a query in the form of a very small subset of items $\mathcal{D}_c \subset \mathcal{D}$. The assumption is that the elements in $\mathcal{D}_c$ are examples of some concept / class / cluster in the data. The algorithm then has to provide a completion to the set $\mathcal{D}_c$—that is, some set $\mathcal{D}'_c \subset \mathcal{D}$ which presumably includes all the elements in $\mathcal{D}_c$ and other elements in $\mathcal{D}$ which are also in this concept / class / cluster[2].

---

[*]ZG is also at CALD, Carnegie Mellon University, Pittsburgh PA 15213.

[1]Google™ Sets is a large-scale clustering algorithm that uses many millions of data instances extracted from web data (Simon Tong, personal communication). We are unable to describe any details of how the algorithm works due its proprietary nature.

[2]From here on, we will use the term "cluster" to refer to the target concept.

We can view this problem from several perspectives. First, the query can be interpreted as elements of some unknown cluster, and the output of the algorithm is the completion of that cluster. Whereas most clustering algorithms are completely unsupervised, here the query provides supervised hints or constraints as to the membership of a particular cluster. We call this view *clustering on demand*, since it involves forming a cluster once some elements of that cluster have been revealed. An important advantage of this approach over traditional clustering is that the few elements in the query can give useful information as to the features which are relevant for forming the cluster. For example, the query "Bush", "Nixon", "Reagan" suggests that the features *republican* and *US President* are relevant to the cluster, while the query "Bush", "Putin", "Blair" suggests that *current* and *world leader* are relevant. Given the huge number of features in many real world data sets, such hints as to feature relevance can produce much more sensible clusters.

Second, we can think of the goal of the algorithm to be to solve a particular *information retrieval* problem [2, 3, 4]. As in other retrieval problems, the output should be relevant to the query, and it makes sense to limit the output to the top few items ranked by relevance to the query. In our experiments, we take this approach and report items ranked by relevance. Our relevance criterion is closely related to a Bayesian framework for understanding patterns of generalization in human cognition [5].

## 2  Bayesian Sets

Let $\mathcal{D}$ be a data set of items, and $\mathbf{x} \in \mathcal{D}$ be an item from this set. Assume the user provides a query set $\mathcal{D}_c$ which is a small subset of $\mathcal{D}$. Our goal is to rank the elements of $\mathcal{D}$ by how well they would "fit into" a set which includes $\mathcal{D}_c$. Intuitively, the task is clear: if the set $\mathcal{D}$ is the set of all movies, and the query set consists of two animated Disney movies, we expect other animated Disney movies to be ranked highly.

We use a model-based probabilistic criterion to measure how well items fit into $\mathcal{D}_c$. Having observed $\mathcal{D}_c$ as belonging to some concept, we want to know how probable it is that $\mathbf{x}$ also belongs with $\mathcal{D}_c$. This is measured by $p(\mathbf{x}|\mathcal{D}_c)$. Ranking items simply by this probability is not sensible since some items may be more probable than others, regardless of $\mathcal{D}_c$. For example, under most sensible models, the probability of a string decreases with the number of characters, the probability of an image decreases with the number of pixels, and the probability of any continuous variable decreases with the precision to which it is measured. We want to remove these effects, so we compute the ratio:

$$\text{score}(\mathbf{x}) = \frac{p(\mathbf{x}|\mathcal{D}_c)}{p(\mathbf{x})} \tag{1}$$

where the denominator is the prior probability of $\mathbf{x}$ and under most sensible models will scale exactly correctly with number of pixels, characters, discretization level, etc. Using Bayes rule, this score can be re-written as:

$$\text{score}(\mathbf{x}) = \frac{p(\mathbf{x}, \mathcal{D}_c)}{p(\mathbf{x})\, p(\mathcal{D}_c)} \tag{2}$$

which can be interpreted as the ratio of the joint probability of observing $\mathbf{x}$ *and* $\mathcal{D}_c$, to the probability of independently observing $\mathbf{x}$ and $\mathcal{D}_c$. Intuitively, this ratio compares the probability that $\mathbf{x}$ and $\mathcal{D}_c$ were generated by the same model with the *same*, though unknown, parameters $\theta$, to the probability that $\mathbf{x}$ and $\mathcal{D}_c$ came from models with *different* parameters $\theta$ and $\theta'$ (see figure 1). Finally, up to a multiplicative constant independent of $\mathbf{x}$, the score can be written as: $\text{score}(\mathbf{x}) = p(\mathcal{D}_c|\mathbf{x})$, which is the probability of observing the query set given $\mathbf{x}$ (i.e. the likelihood of $\mathbf{x}$).

From the above discussion, it is still not clear how one would compute quantities such as $p(\mathbf{x}|\mathcal{D}_c)$ and $p(\mathbf{x})$. A natural model-based way of defining a cluster is to assume that

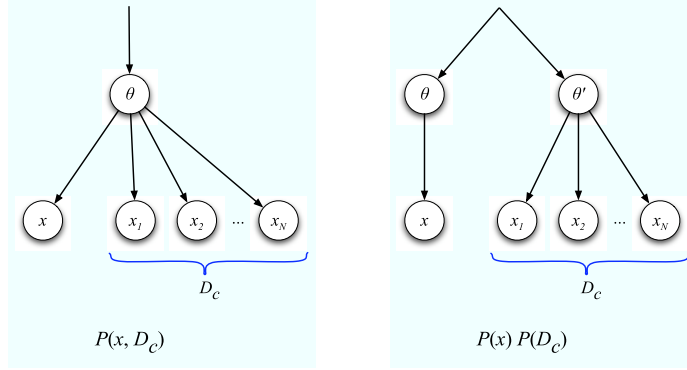

Figure 1: Our Bayesian score compares the hypotheses that the data was generated by each of the above graphical models.

the data points in the cluster all come independently and identically distributed from some simple parameterized statistical model. Assume that the parameterized model is $p(\mathbf{x}|\theta)$ where $\theta$ are the parameters. If the data points in $\mathcal{D}_c$ all belong to one cluster, then under this definition they were generated from the same setting of the parameters; however, that setting is unknown, so we need to average over possible parameter values weighted by some prior density on parameter values, $p(\theta)$. Using these considerations and the basic rules of probability we arrive at:

$$p(\mathbf{x}) = \int p(\mathbf{x}|\theta)\, p(\theta)\, d\theta \tag{3}$$

$$p(\mathcal{D}_c) = \int \prod_{\mathbf{x}_i \in \mathcal{D}_c} p(\mathbf{x}_i|\theta)\, p(\theta)\, d\theta \tag{4}$$

$$p(\mathbf{x}|\mathcal{D}_c) = \int p(\mathbf{x}|\theta)\, p(\theta|\mathcal{D}_c)\, d\theta \tag{5}$$

$$p(\theta|\mathcal{D}_c) = \frac{p(\mathcal{D}_c|\theta)\, p(\theta)}{p(\mathcal{D}_c)} \tag{6}$$

We are now fully equipped to describe the "Bayesian Sets" algorithm:

---
**Bayesian Sets Algorithm**

> **background:** a set of items $\mathcal{D}$, a probabilistic model $p(\mathbf{x}|\theta)$ where
> $\quad$ $\mathbf{x} \in \mathcal{D}$, a prior on the model parameters $p(\theta)$
> **input:** a query $\mathcal{D}_c = \{\mathbf{x}_i\} \subset \mathcal{D}$
> **for all $\mathbf{x} \in \mathcal{D}$ do**
> $\quad$ compute $\quad$ $\text{score}(\mathbf{x}) = \dfrac{p(\mathbf{x}|\mathcal{D}_c)}{p(\mathbf{x})}$
> **end for**
> **output:** return elements of $\mathcal{D}$ sorted by decreasing score

---

We mention two properties of this algorithm to assuage two common worries with Bayesian methods—tractability and sensitivity to priors:

1. For the simple models we will consider, the integrals (3)-(5) are analytical. In fact, for the model we consider in section 3 computing all the scores can be reduced to a single sparse matrix multiplication.

2. Although it clearly makes sense to put some thought into choosing sensible models $p(\mathbf{x}|\theta)$ and priors $p(\theta)$, we will show in 5 that even with very simple models and almost no tuning of the prior one can get very competitive retrieval results. In practice, we use a simple empirical heuristic which sets the prior to be vague but centered on the mean of the data in $\mathcal{D}$.

## 3  Sparse Binary Data

We now derive in more detail the application of the Bayesian Sets algorithm to sparse binary data. This type of data is a very natural representation for the large datasets we used in our evaluations (section 5). Applications of Bayesian Sets to other forms of data (real-valued, discrete, ordinal, strings) are also possible, and especially practical if the statistical model is a member of the exponential family (section 4).

Assume each item $\mathbf{x}_i \in \mathcal{D}_c$ is a binary vector $\mathbf{x}_i = (x_{i1}, \dots, x_{iJ})$ where $x_{ij} \in \{0,1\}$, and that each element of $\mathbf{x}_i$ has an independent Bernoulli distribution:

$$p(\mathbf{x}_i|\theta) = \prod_{j=1}^{J} \theta_j^{x_{ij}} (1-\theta_j)^{1-x_{ij}} \tag{7}$$

The conjugate prior for the parameters of a Bernoulli distribution is the Beta distribution:

$$p(\theta|\alpha,\beta) = \prod_{j=1}^{J} \frac{\Gamma(\alpha_j+\beta_j)}{\Gamma(\alpha_j)\Gamma(\beta_j)} \, \theta_j^{\alpha_j-1} (1-\theta_j)^{\beta_j-1} \tag{8}$$

where $\alpha$ and $\beta$ are hyperparameters, and the Gamma function is a generalization of the factorial function. For a query $\mathcal{D}_c = \{\mathbf{x}_i\}$ consisting of $N$ vectors it is easy to show that:

$$p(\mathcal{D}_c|\alpha,\beta) = \prod_{j} \frac{\Gamma(\alpha_j+\beta_j)}{\Gamma(\alpha_j)\Gamma(\beta_j)} \frac{\Gamma(\tilde{\alpha}_j)\Gamma(\tilde{\beta}_j)}{\Gamma(\tilde{\alpha}_j+\tilde{\beta}_j)} \tag{9}$$

where $\tilde{\alpha} = \alpha + \sum_{i=1}^{N} x_{ij}$ and $\tilde{\beta} = \beta + N - \sum_{i=1}^{N} x_{ij}$. For an item $\mathbf{x} = (x_{.1} \dots x_{.J})$ the score, written with the hyperparameters explicit, can be computed as follows:

$$\text{score}(\mathbf{x}) = \frac{p(\mathbf{x}|\mathcal{D}_c,\alpha,\beta)}{p(\mathbf{x}|\alpha,\beta)} = \prod_{j} \frac{\frac{\Gamma(\alpha_j+\beta_j+N)}{\Gamma(\alpha_j+\beta_j+N+1)} \frac{\Gamma(\tilde{\alpha}_j+x_{.j})\Gamma(\tilde{\beta}_j+1-x_{.j})}{\Gamma(\tilde{\alpha}_j)\Gamma(\tilde{\beta}_j)}}{\frac{\Gamma(\alpha_j+\beta_j)}{\Gamma(\alpha_j+\beta_j+1)} \frac{\Gamma(\alpha_j+x_{.j})\Gamma(\beta_j+1-x_{.j})}{\Gamma(\alpha_j)\Gamma(\beta_j)}} \tag{10}$$

This daunting expression can be dramatically simplified. We use the fact that $\Gamma(x) = (x-1)\,\Gamma(x-1)$ for $x > 1$. For each $j$ we can consider the two cases $x_{.j} = 0$ and $x_{.j} = 1$ and separately. For $x_{.j} = 1$ we have a contribution $\frac{\alpha_j+\beta_j}{\alpha_j+\beta_j+N} \frac{\tilde{\alpha}_j}{\alpha_j}$. For $x_{.j} = 0$ we have a contribution $\frac{\alpha_j+\beta_j}{\alpha_j+\beta_j+N} \frac{\tilde{\beta}_j}{\beta_j}$. Putting these together we get:

$$\text{score}(\mathbf{x}) = \prod_{j} \frac{\alpha_j+\beta_j}{\alpha_j+\beta_j+N} \left(\frac{\tilde{\alpha}_j}{\alpha_j}\right)^{x_{.j}} \left(\frac{\tilde{\beta}_j}{\beta_j}\right)^{1-x_{.j}} \tag{11}$$

The log of the score is *linear* in $\mathbf{x}$:

$$\log \text{score}(\mathbf{x}) = c + \sum_{j} q_j x_{.j} \tag{12}$$

where

$$c = \sum_{j} \log(\alpha_j+\beta_j) - \log(\alpha_j+\beta_j+N) + \log \tilde{\beta}_j - \log \beta_j \tag{13}$$

and
$$q_j = \log \tilde{\alpha}_j - \log \alpha_j - \log \tilde{\beta}_j + \log \beta_j \tag{14}$$

If we put the entire data set $\mathcal{D}$ into one large matrix $\mathbf{X}$ with $J$ columns, we can compute the vector $\mathbf{s}$ of log scores for all points using a single matrix vector multiplication

$$\mathbf{s} = c + \mathbf{Xq} \tag{15}$$

For sparse data sets this linear operation can be implemented very efficiently. Each query $\mathcal{D}_c$ corresponds to computing the vector $\mathbf{q}$ and scalar $c$. This can also be done efficiently if the query is also sparse, since most elements of $\mathbf{q}$ will equal $\log \beta_j - \log(\beta_j + N)$ which is independent of the query.

## 4  Exponential Families

We generalize the above result to models in the exponential family. The distribution for such models can be written in the form $p(\mathbf{x}|\theta) = f(\mathbf{x})g(\theta)\exp\{\theta^\top \mathbf{u}(\mathbf{x})\}$, where $\mathbf{u}(\mathbf{x})$ is a $K$-dimensional vector of sufficient statistics, $\theta$ are the natural parameters, and $f$ and $g$ are non-negative functions. The conjugate prior is $p(\theta|\eta, \nu) = h(\eta, \nu)g(\theta)^\eta \exp\{\theta^\top \nu\}$, where $\eta$ and $\nu$ are hyperparameters, and $h$ normalizes the distribution.

Given a query $\mathcal{D}_c = \{\mathbf{x}_i\}$ with $N$ items, and a candidate $\mathbf{x}$, it is not hard to show that the score for the candidate is:

$$\text{score}(\mathbf{x}) = \frac{h(\eta + 1, \nu + \mathbf{u}(\mathbf{x}))\, h(\eta + N, \nu + \sum_i \mathbf{u}(\mathbf{x}_i))}{h(\eta, \nu)\, h(\eta + N + 1, \nu + \mathbf{u}(\mathbf{x}) + \sum_i \mathbf{u}(\mathbf{x}_i))} \tag{16}$$

This expression helps us understand when the score can be computed efficiently. First of all, the score only depends on the size of the query ($N$), the sufficient statistics computed from each candidate, and from the whole query. It therefore makes sense to precompute $\mathbf{U}$, a matrix of sufficient statistics corresponding to $\mathbf{X}$. Second, whether the score is a linear operation on $\mathbf{U}$ depends on whether $\log h$ is linear in the second argument. This is the case for the Bernoulli distribution, but not for all exponential family distributions. However, for many distributions, such as diagonal covariance Gaussians, even though the score is nonlinear in $\mathbf{U}$, it can be computed by applying the nonlinearity elementwise to $\mathbf{U}$. For sparse matrices, the score can therefore still be computed in time linear in the number of non-zero elements of $\mathbf{U}$.

## 5  Results

We ran our Bayesian Sets algorithm on three different datasets: the Groliers Encyclopedia dataset, consisting of the text of the articles in the Encyclopedia, the EachMovie dataset, consisting of movie ratings by users of the EachMovie service, and the NIPS authors dataset, consisting of the text of articles published in NIPS volumes 0-12 (spanning the 1987-1999 conferences). The Groliers dataset is 30991 articles by 15276 words, where the entries are the number of times each word appears in each document. We preprocess (binarize) the data by column normalizing each word, and then thresholding so that a (article,word) entry is 1 if that word has a frequency of more than twice the article mean. We do essentially no tuning of the hyperparameters. We use broad empirical priors, where $\alpha = c \times \mathbf{m}$, $\beta = c \times (1 - \mathbf{m})$ where $\mathbf{m}$ is a mean vector over all articles, and $c = 2$. The analogous priors are used for both other datasets.

The EachMovie dataset was preprocessed, first by removing movies rated by less than 15 people, and people who rated less than 200 movies. Then the dataset was binarized so that a (person, movie) entry had value 1 if the person gave the movie a rating above 3 stars (from a possible 0-5 stars). The data was then column normalized to account for overall movie popularity. The size of the dataset after preprocessing was 1813 people by 1532 movies.

Finally the NIPS author dataset (13649 words by 2037 authors), was preprocessed very similarly to the Grolier dataset. It was binarized by column normalizing each author, and then thresholding so that a (word,author) entry is 1 if the author uses that word more frequently than twice the word mean across all authors.

The results of our experiments, and comparisons with Google Sets for word and movie queries are given in tables 2 and 3. Unfortunately, NIPS authors have not yet achieved the kind of popularity on the web necessary for Google Sets to work effectively. Instead we list the top words associated with the cluster of authors given by our algorithm (table 4).

The running times of our algorithm on all three datasets are given in table 1. All experiments were run in Matlab on a 2GHz Pentium 4, Toshiba laptop. Our algorithm is very fast both at pre-processing the data, and answering queries (about 1 sec per query).

|  | GROLIERS | EACHMOVIE | NIPS |
|---|---|---|---|
| SIZE | $30991 \times 15276$ | $1813 \times 1532$ | $13649 \times 2037$ |
| NON-ZERO ELEMENTS | 2,363,514 | 517,709 | 933,295 |
| PREPROCESS TIME | 6.1s | 0.56s | 3.22s |
| QUERY TIME | 1.1s | 0.34s | 0.47s |

Table 1: For each dataset we give the size of that dataset along with the time taken to do the (one-time) preprocessing and the time taken to make a query (both in seconds).

| QUERY: WARRIOR, SOLDIER | | QUERY: ANIMAL | | QUERY: FISH, WATER, CORAL | |
|---|---|---|---|---|---|
| GOOGLE SETS | BAYES SETS | GOOGLE SETS | BAYES SETS | GOOGLE SETS | BAYES SETS |
| WARRIOR | SOLDIER | ANIMAL | ANIMAL | FISH | WATER |
| SOLDIER | WARRIOR | PLANT | ANIMALS | WATER | FISH |
| SPY | MERCENARY | FREE | PLANT | CORAL | SURFACE |
| ENGINEER | CAVALRY | LEGAL | HUMANS | AGRICULTURE | SPECIES |
| MEDIC | BRIGADE | FUNGAL | FOOD | FOREST | WATERS |
| SNIPER | COMMANDING | HUMAN | SPECIES | RICE | MARINE |
| DEMOMAN | SAMURAI | HYSTERIA | MAMMALS | SILK ROAD | FOOD |
| PYRO | BRIGADIER | VEGETABLE | AGO | RELIGION | TEMPERATURE |
| SCOUT | INFANTRY | MINERAL | ORGANISMS | HISTORY POLITICS | OCEAN |
| PYROMANIAC | COLONEL | INDETERMINATE | VEGETATION | DESERT | SHALLOW |
| HWGUY | SHOGUNATE | FOZZIE BEAR | PLANTS | ARTS | FT |

Table 2: Clusters of words found by Google Sets and Bayesian Sets based on the given queries. The top few are shown for each query and each algorithm. Bayesian Sets was run using Grolier Encyclopedia data.

It is very difficult to objectively evaluate our results since there is no ground truth for this task. One person's idea of a good query cluster may differ drastically from another person's. We chose to compare our algorithm to Google Sets since it was our main inspiration and it is currently the most public and commonly used algorithm for performing this task.

Since we do not have access to the Google Sets algorithm it was impossible for us to run their method on our datasets. Moreover, Google Sets relies on vast amounts of web data, which we do not have. Despite those two important caveats, Google Sets clearly "knows" a lot about movies[3] and words, and the comparison to Bayesian Sets is informative.

We found that Google Sets performed very well when the query consisted of items which can be found listed on the web (e.g. Cambridge colleges). On the other hand, for more abstract concepts (e.g. "soldier" and "warrior", see Table 2) our algorithm returned more sensible completions.

While we believe that most of our results are self-explanatory, there are a few details that we would like to elaborate on. The top query in table 3 consists of two classic romantic movies,

| QUERY: GONE WITH THE WIND, CASABLANCA | |
| --- | --- |
| GOOGLE SETS | BAYES SETS |
| CASABLANCA (1942) | GONE WITH THE WIND (1939) |
| GONE WITH THE WIND (1939) | CASABLANCA (1942) |
| ERNEST SAVES CHRISTMAS (1988) | THE AFRICAN QUEEN (1951) |
| CITIZEN KANE (1941) | THE PHILADELPHIA STORY (1940) |
| PET DETECTIVE (1994) | MY FAIR LADY (1964) |
| VACATION (1983) | THE ADVENTURES OF ROBIN HOOD (1938) |
| WIZARD OF OZ (1939) | THE MALTESE FALCON (1941) |
| THE GODFATHER (1972) | REBECCA (1940) |
| LAWRENCE OF ARABIA (1962) | SINGING IN THE RAIN (1952) |
| ON THE WATERFRONT (1954) | IT HAPPENED ONE NIGHT (1934) |

| QUERY: MARY POPPINS, TOY STORY | | QUERY: CUTTHROAT ISLAND, LAST ACTION HERO | |
| --- | --- | --- | --- |
| GOOGLE SETS | BAYES SETS | GOOGLE SETS | BAYES SETS |
| TOY STORY | MARY POPPINS | LAST ACTION HERO | CUTTHROAT ISLAND |
| MARY POPPINS | TOY STORY | CUTTHROAT ISLAND | LAST ACTION HERO |
| TOY STORY 2 | WINNIE THE POOH | GIRL | KULL THE CONQUEROR |
| MOULIN ROUGE | CINDERELLA | END OF DAYS | VAMPIRE IN BROOKLYN |
| THE FAST AND THE FURIOUS | THE LOVE BUG | HOOK | SPRUNG |
| PRESQUE RIEN | BEDKNOBS AND BROOMSTICKS | THE COLOR OF NIGHT | JUDGE DREDD |
| SPACED | DAVY CROCKETT | CONEHEADS | WILD BILL |
| BUT I'M A CHEERLEADER | THE PARENT TRAP | ADDAMS FAMILY I | HIGHLANDER III |
| MULAN | DUMBO | ADDAMS FAMILY II | VILLAGE OF THE DAMNED |
| WHO FRAMED ROGER RABBIT | THE SOUND OF MUSIC | SINGLES | FAIR GAME |

Table 3: Clusters of movies found by Google Sets and Bayesian Sets based on the given queries. The top 10 are shown for each query and each algorithm. Bayesian Sets was run using the EachMovie dataset.

and while most of the movies returned by Bayesian Sets are also classic romances, hardly any of the movies returned by Google Sets are romances, and it would be difficult to call "Ernest Saves Christmas" either a romance or a classic. Both "Cutthroat Island" and "Last Action Hero" are action movie flops, as are many of the movies given by our algorithm for that query. All the Bayes Sets movies associated with the query "Mary Poppins" and "Toy Story" are children's movies, while 5 of Google Sets' movies are not. "But I'm a Cheerleader", while appearing to be a children's movie, is actually an R rated movie involving lesbian and gay teens.

| QUERY: A.SMOLA, B.SCHOLKOPF | | QUERY: L.SAUL, T.JAAKKOLA | | QUERY: A.NG, R.SUTTON | |
| --- | --- | --- | --- | --- | --- |
| TOP MEMBERS | TOP WORDS | TOP MEMBERS | TOP WORDS | TOP MEMBERS | TOP WORDS |
| A.SMOLA | VECTOR | L.SAUL | LOG | R.SUTTON | DECISION |
| B.SCHOLKOPF | SUPPORT | T.JAAKKOLA | LIKELIHOOD | A.NG | REINFORCEMENT |
| S.MIKA | KERNEL | M.RAHIM | MODELS | Y.MANSOUR | ACTIONS |
| G.RATSCH | PAGES | M.JORDAN | MIXTURE | B.RAVINDRAN | REWARDS |
| R.WILLIAMSON | MACHINES | N.LAWRENCE | CONDITIONAL | D.KOLLER | REWARD |
| K.MULLER | QUADRATIC | T.JEBARA | PROBABILISTIC | D.PRECUP | START |
| J.WESTON | SOLVE | W.WIEGERINCK | EXPECTATION | C.WATKINS | RETURN |
| J.SHAWE-TAYLOR | REGULARIZATION | M.MEILA | PARAMETERS | R.MOLL | RECEIVED |
| V.VAPNIK | MINIMIZING | S.IKEDA | DISTRIBUTION | T.PERKINS | MDP |
| T.ONODA | MIN | D.HAUSSLER | ESTIMATION | D.MCALLESTER | SELECTS |

Table 4: NIPS authors found by Bayesian Sets based on the given queries. The top 10 are shown for each query along with the top 10 words associated with that cluster of authors. Bayesian Sets was run using NIPS data from vol 0-12 (1987-1999 conferences).

The NIPS author dataset is rather small, and co-authors of NIPS papers appear very similar to each other. Therefore, many of the authors found by our algorithm are co-authors of a NIPS paper with one or more of the query authors. An example where this is not the case is Wim Wiegerinck, who we do not believe ever published a NIPS paper with Lawrence Saul or Tommi Jaakkola, though he did have a NIPS paper on variational learning and graphical models.

As part of the evaluation of our algorithm, we showed 30 naïve subjects the unlabeled results of Bayesian Sets and Google Sets for the queries shown from the EachMovie and Groliers Encyclopedia datasets, and asked them to choose which they preferred. The results of this study are given in table 5.

| Query | % Bayes Sets | P-value |
|---|---|---|
| Warrior | 96.7 | < 0.0001 |
| Animal | 93.3 | < 0.0001 |
| Fish | 90.0 | < 0.0001 |
| Gone with the Wind | 86.7 | < 0.0001 |
| Mary Poppins | 96.7 | < 0.0001 |
| Cutthroat Island | 81.5 | 0.0008 |

Table 5: For each evaluated query (listed by first query item), we give the percentage of respondents who preferred the results given by Bayesian Sets and the p-value rejecting the null hypothesis that Google Sets is preferable to Bayesian Sets on that particular query.

Since, in the case of binary data, our method reduces to a matrix-vector multiplication, we also came up with ten heuristic matrix-vector methods which we ran on the same queries, using the same datasets. Descriptions and results can be found in supplemental material on the authors websites.

## 6   Conclusions

We have described an algorithm which takes a query consisting of a small set of items, and returns additional items which belong in this set. Our algorithm computes a score for each item by comparing the posterior probability of that item given the set, to the prior probability of that item. These probabilities are computed with respect to a statistical model for the data, and since the parameters of this model are unknown they are marginalized out.

For exponential family models with conjugate priors, our score can be computed exactly and efficiently. In fact, we show that for sparse binary data, scoring all items in a large data set can be accomplished using a single sparse matrix-vector multiplication. Thus, we get a very fast and practical Bayesian algorithm without needing to resort to approximate inference. For example, a sparse data set with over 2 million nonzero entries (Grolier) can be queried in just over 1 second.

Our method does well when compared to Google Sets in terms of set completions, demonstrating that this Bayesian criterion can be useful in realistic problem domains. One of the problems we have not yet addressed is deciding on the size of the response set. Since the scores have a probabilistic interpretation, it should be possible to find a suitable threshold to these probabilities. In the future, we will incorporate such a threshold into our algorithm.

The problem of retrieving sets of items is clearly relevant to many application domains. Our algorithm is very flexible in that it can be combined with a wide variety of types of data (e.g. sequences, images, etc.) and probabilistic models. We plan to explore efficient implementations of some of these extensions. We believe that with even larger datasets the Bayesian Sets algorithm will be a very useful tool for many application areas.

**Acknowledgements:** Thanks to Avrim Blum and Simon Tong for useful discussions, and to Sam Roweis for some of the data. ZG was partially supported at CMU by the DARPA CALO project.

## Footnotes

[3]In fact, one of the example queries on the Google Sets website is a query of movie titles.

## References

[1] Google <sup>TM</sup> Sets. http://labs.google.com/sets

[2] Lafferty, J. and Zhai, C. (2002) Probabilistic relevance models based on document and query generation. In *Language modeling and information retrieval*.

[3] Ponte, J. and Croft, W. (1998) A language modeling approach to information retrieval. *SIGIR*.

[4] Robertson, S. and Sparck Jones, K. (1976). Relevance weighting of search terms. *J Am Soc Info Sci*.

[5] Tenenbaum, J. B. and Griffiths, T. L. (2001). Generalization, similarity, and Bayesian inference. *Behavioral and Brain Sciences*, 24:629–641.

[6] Tong, S. (2005). Personal communication.